# Infinite Latent SVM for Classification and Multi-task Learning

**Jun Zhu[†], Ning Chen[†], and Eric P. Xing[‡]**
[†]Dept. of Computer Science & Tech., TNList Lab, Tsinghua University, Beijing 100084, China
[‡]Machine Learning Department, Carnegie Mellon University, Pittsburgh, PA 15213, USA
dcszj@tsinghua.edu.cn;chenn07@mails.thu.edu.cn;epxing@cs.cmu.edu

## Abstract

Unlike existing nonparametric Bayesian models, which rely solely on specially conceived priors to incorporate domain knowledge for discovering improved latent representations, we study nonparametric Bayesian inference with regularization on the desired posterior distributions. While priors can indirectly affect posterior distributions through Bayes' theorem, imposing posterior regularization is arguably more direct and in some cases can be much easier. We particularly focus on developing *infinite latent support vector machines* (iLSVM) and *multi-task infinite latent support vector machines* (MT-iLSVM), which explore the large-margin idea in combination with a nonparametric Bayesian model for discovering predictive latent features for classification and multi-task learning, respectively. We present efficient inference methods and report empirical studies on several benchmark datasets. Our results appear to demonstrate the merits inherited from both large-margin learning and Bayesian nonparametrics.

## 1 Introduction

Nonparametric Bayesian latent variable models have recently gained remarkable popularity in statistics and machine learning, partly owning to their desirable "nonparametric" nature which allows practitioners to "sidestep" the difficult model selection problem, e.g., figuring out the unknown number of components (or classes) in a mixture model [2] or determining the unknown dimensionality of latent features [12], by using an appropriate prior distribution with a large support. Among the most commonly used priors are Gaussian process (GP) [24], Dirichlet process (DP) [2] and Indian buffet process (IBP) [12].

However, standard nonparametric Bayesian models are limited in that they usually make very strict and unrealistic assumptions on data, such as that observations being homogeneous or exchangeable. A number of recent developments in Bayesian nonparametrics have attempted to alleviate such limitations. For example, to handle heterogenous observations, predictor-dependent processes [20] have been proposed; and to relax the exchangeability assumption, various correlation structures, such as hierarchical structures [26], temporal or spatial dependencies [5], and stochastic ordering dependencies [13, 10], have been introduced. However, all these methods rely solely on crafting a nonparametric Bayesian prior encoding some special structure, which can *indirectly* influence the posterior distribution of interest via trading-off with likelihood models. Since it is the posterior distributions, which capture the latent structures to be learned, that are of our ultimate interest, an arguably more *direct* way to learn a desirable latent-variable model is to impose posterior regularization (i.e., regularization on posterior distributions), as we will explore in this paper. Another reason for using posterior regularization is that in some cases it is more natural and easier to incorporate domain knowledge, such as the large-margin [15, 31] or manifold constraints [14], directly on posterior distributions rather than through priors, as shown in this paper.

Posterior regularization, usually through imposing constraints on the posterior distributions of latent variables or via some information projection, has been widely studied in learning a finite log-linear model from partially observed data, including generalized expectation [21], posterior regulariza-

tion [11], and alternating projection [6], all of which are doing maximum likelihood estimation (MLE) to learn a single set of model parameters by optimizing an objective. Recent attempts toward learning a posterior distribution of model parameters include the "learning from measurements" [19], maximum entropy discrimination [15] and MedLDA [31]. But again, all these methods are limited to finite parametric models. To our knowledge, very few attempts have been made to impose posterior regularization on nonparametric Bayesian latent variable models. One exception is our recent work of infinite SVM (iSVM) [32], a DP mixture of large-margin classifiers. iSVM is a latent class model that assigns each data example to a single mixture component for classification and the unknown number of mixture components is automatically resolved from data.

In this paper, we present a general formulation of performing nonparametric Bayesian inference subject to appropriate posterior constraints. In particular, we concentrate on developing the *infinite latent support vector machines* (iLSVM) and *multi-task infinite latent support vector machines* (MT-iLSVM), which explore the discriminative large-margin idea to learn infinite latent feature models for classification and multi-task learning [3, 4], respectively. As such, our methods as well as [32] represent an attempt to push forward the interface between Bayesian nonparametrics and large margin learning, which have complementary advantages but have been largely treated as two separate subfields in the machine learning community. Technically, although it is intuitively natural for MLE-based methods to include a regularization term on the posterior distributions of latent variables, this is not straightforward for Bayesian inference because we do not have an optimization objective to be regularized. We base our work on the interpretation of the Bayes' theorem by Zellner [29], namely, the Bayes' theorem can be reformulated as a minimization problem. Under this optimization framework, we incorporate posterior constraints to do *regularized Bayesian inference*, with a penalty term that measures the violation of the constraints. Both iLSVM and MT-iLSVM are special cases that explore the large-margin principle to consider supervising information for learning predictive latent features, which are good for classification or multi-task learning. We use the nonparametric IBP prior to allow the models to have an unbounded number of latent features. The regularized inference problem can be efficiently solved with an iterative procedure, which leverages existing high-performance convex optimization techniques.

**Related Work**: As stated above, both iLSVM and MT-iLSVM generalize the ideas of iSVM to infinite latent feature models. For multi-task learning, nonparametric Bayesian models have been developed in [28, 23] for learning features shared by multiple tasks. But these methods are based on standard Bayesian inference, without the ability to consider posterior regularization, such as the large-margin constraints or the manifold constraints [14]. Finally, MT-iLSVM is a nonparametric Bayesian generalization of the popular multi-task learning methods [1, 16], as explained shortly.

## 2  Regularized Bayesian Inference with Posterior Constraints

In this section, we present the general framework of regularized Bayesian inference with posterior constraints. We begin with a brief review of the basic results due to Zellner [29].

### 2.1  Bayesian Inference as a Learning Model

Let $\mathbb{M}$ be a model space, containing any variables whose posterior distributions we are trying to infer. Bayesian inference starts with a prior distribution $\pi(\mathcal{M})$ and a likelihood function $p(\mathbf{x}|\mathcal{M})$ indexed by the model $\mathcal{M} \in \mathbb{M}$. Then, by the Bayes' theorem, the posterior distribution is

$$p(\mathcal{M}|\mathbf{x}_1, \cdots, \mathbf{x}_N) = \frac{\pi(\mathcal{M}) \prod_{n=1}^N p(\mathbf{x}_n|\mathcal{M})}{p(\mathbf{x}_1, \cdots, \mathbf{x}_N)}, \tag{1}$$

where $p(\mathbf{x}_1, \cdots, \mathbf{x}_N)$ is the marginal likelihood or evidence of observed data. Zellner [29] first showed that the posterior distribution due to the Bayes' theorem is the solution of the problem

$$\min_{p(\mathcal{M})} \quad \mathrm{KL}(p(\mathcal{M})\|\pi(\mathcal{M})) - \sum_{n=1}^N \int \log p(\mathbf{x}_n|\mathcal{M})p(\mathcal{M})d\mathcal{M} \tag{2}$$

$$\mathrm{s.t.}: \quad p(\mathcal{M}) \in \mathcal{P}_{\mathrm{prob}},$$

where $\mathrm{KL}(p(\mathcal{M})\|\pi(\mathcal{M}))$ is the Kullback-Leibler (KL) divergence, and $\mathcal{P}_{\mathrm{prob}}$ is the space of valid probability distributions with an appropriate dimension.

### 2.2  Regularized Bayesian Inference with Posterior Constraints

As commented by E.T. Jaynes [29], "this fresh interpretation of Bayes' theorem could make the use of Bayesian methods more attractive and widespread, and stimulate new developments in

the general theory of inference". Below, we study how to extend the basic results to incorporate posterior constraints in Bayesian inference. In the standard Bayesian inference, the constraints (i.e., $p(\mathcal{M}) \in \mathcal{P}_{\text{prob}}$) do not have auxiliary free parameters. In general, *regularized Bayesian inference* solves the constrained optimization problem

$$\min_{p(\mathcal{M}), \boldsymbol{\xi}} \quad \text{KL}(p(\mathcal{M}) \| \pi(\mathcal{M})) - \sum_{n=1}^{N} \int \log p(\mathbf{x}_n | \mathcal{M}) p(\mathcal{M}) d\mathcal{M} + U(\boldsymbol{\xi}) \tag{3}$$

$$\text{s.t.:} \quad p(\mathcal{M}) \in \mathcal{P}_{\text{post}}(\boldsymbol{\xi}),$$

where $\mathcal{P}_{\text{post}}(\boldsymbol{\xi})$ is a subspace of distributions that satisfy a set of constraints. The auxiliary parameters $\boldsymbol{\xi}$ are usually nonnegative and interpreted as slack variables. $U(\boldsymbol{\xi})$ is a convex function, which usually corresponds to a surrogate loss (e.g., hinge loss) of a prediction rule, as we shall see.

We can use an iterative procedure to do the regularized Bayesian inference based on convex optimization techniques. The general recipe is that we use the Lagrangian method by introducing Lagrangian multipliers $\boldsymbol{\omega}$. Then, we iteratively solve for $p(\mathcal{M})$ with $\boldsymbol{\omega}$ and $\boldsymbol{\xi}$ fixed; and solve for $\boldsymbol{\omega}$ and $\boldsymbol{\xi}$ with $p(\mathcal{M})$ given. For the first step, we can use sampling or variational methods [9] to do approximate inference; and under certain conditions, such as using the constraints based on posterior expectation [21], the second step can be efficiently done using high-performance convex optimization techniques, as we shall see.

## 3 Infinite Latent Support Vector Machines

In this section, we concretize the ideas of regularized Bayesian inference by particularly focusing on developing large-margin classifiers with an unbounded dimension of latent features, which can be used as a representation of examples for the single-task classification or as a common representation that captures relationships among multiple tasks for multi-task learning.

We first present the single-task classification model. The basic setup is that we project each data example $\mathbf{x} \in \mathcal{X} \subset \mathbb{R}^D$ to a latent feature vector $\mathbf{z}$. Here, we consider binary features[1]. Given a set of $N$ data examples, let $\mathbf{Z}$ be the matrix, of which each row is a binary vector $\mathbf{z}_n$ associated with data sample $n$. Instead of pre-specifying a fixed dimension of $\mathbf{z}$, we resort to the nonparametric Bayesian methods and let $\mathbf{z}$ have an infinite number of dimensions. To make the expected number of active latent features finite, we put the well-studied IBP prior on the binary feature matrix $\mathbf{Z}$.

### 3.1 Indian Buffet Process

Indian buffet process (IBP) was proposed in [12] and has been successfully applied in various fields, such as link prediction [22] and multi-task learning [23]. We focus on its stick-breaking construction [25], which is good for developing efficient inference methods. Let $\pi_k \in (0, 1)$ be a parameter associated with column $k$ of the binary matrix $\mathbf{Z}$. Given $\pi_k$, each $z_{nk}$ in column $k$ is sampled independently from $\text{Bernoulli}(\pi_k)$. The parameters $\boldsymbol{\pi}$ are generated by a stick-breaking process

$$\pi_1 = \nu_1, \text{ and } \pi_k = \nu_k \pi_{k-1} = \prod_{i=1}^{k} \nu_i, \tag{4}$$

where $\nu_i \sim \text{Beta}(\alpha, 1)$. This process results in a decreasing sequence of probabilities $\pi_k$. Specifically, given a finite dataset, the probability of seeing feature $k$ decreases exponentially with $k$.

### 3.2 Infinite Latent Support Vector Machines

We consider the multi-way classification, where each training data is provided with a categorical label $y$, where $y \in \mathcal{Y} \stackrel{\text{def}}{=} \{1, \cdots, L\}$. For binary classification and regression, similar procedure can be applied to impose large-margin constraints on posterior distributions. Suppose that the latent features $\mathbf{z}$ are given, then we can define the *latent discriminant function* as

$$f(y, \mathbf{x}, \mathbf{z}; \boldsymbol{\eta}) \stackrel{\text{def}}{=} \boldsymbol{\eta}^\top \mathbf{g}(y, \mathbf{x}, \mathbf{z}), \tag{5}$$

where $\mathbf{g}(y, \mathbf{x}, \mathbf{z})$ is a vector stacking of $L$ subvectors[2] of which the $y$th is $\mathbf{z}^\top$ and all the others are zero. Since we are doing Bayesian inference, we need to maintain the entire distribution profile of

the latent features $\mathbf{Z}$. However, in order to make a prediction on the observed data $\mathbf{x}$, we need to get rid of the uncertainty of $\mathbf{Z}$. Here, we define the *effective discriminant function* as an expectation[3] (i.e., a weighted average considering all possible values of $\mathbf{Z}$) of the latent discriminant function. To make the model fully Bayesian, we also treat $\boldsymbol{\eta}$ as random and aim to infer the posterior distribution $p(\mathbf{Z}, \boldsymbol{\eta})$ from given data. More formally, the effective discriminant function $f : \mathcal{X} \times \mathcal{Y} \mapsto \mathbb{R}$ is

$$f(y, \mathbf{x}; p(\mathbf{Z}, \boldsymbol{\eta})) \stackrel{\text{def}}{=} \mathbb{E}_{p(\mathbf{Z}, \boldsymbol{\eta})}[f(y, \mathbf{x}, \mathbf{z}; \boldsymbol{\eta})] = \mathbb{E}_{p(\mathbf{Z}, \boldsymbol{\eta})}[\boldsymbol{\eta}^\top \mathbf{g}(y, \mathbf{x}, \mathbf{z})]. \tag{6}$$

Note that although the number of latent features is allowed to be infinite, with probability one, the number of non-zero features is finite when only a finite number of data are observed, under the IBP prior. Moreover, to make it computationally feasible, we usually set a finite upper bound $K$ to the number of possible features, where $K$ is sufficiently large and known as the truncation level (See Sec 3.4 and Appendix A.2 for details). As shown in [9], the $\ell_1$-distance truncation error of marginal distributions decreases exponentially as $K$ increases.

With the above definitions, we define the $\mathcal{P}_{\text{post}}(\boldsymbol{\xi})$ in problem (3) using large-margin constraints as

$$\mathcal{P}_{\text{post}}^c(\boldsymbol{\xi}) \stackrel{\text{def}}{=} \left\{ p(\mathbf{Z}, \boldsymbol{\eta}) \middle| \begin{array}{l} \forall n \in \mathcal{I}_{\text{tr}} : f(y_n, \mathbf{x}_n; p(\mathbf{Z}, \boldsymbol{\eta})) - f(y, \mathbf{x}_n; p(\mathbf{Z}, \boldsymbol{\eta})) \geq \ell(y, y_n) - \xi_n, \forall y \\ \xi_n \geq 0 \end{array} \right\} \tag{7}$$

and define the penalty function as $U^c(\boldsymbol{\xi}) \stackrel{\text{def}}{=} C \sum_{n \in \mathcal{I}_{\text{tr}}} \xi_n^p$, where $p \geq 1$. If $p$ is 1, minimizing $U^c(\boldsymbol{\xi})$ is equivalent to minimizing the hinge-loss (or $\ell_1$-loss) $\mathcal{R}_h^c$ of the prediction rule (9), where $\mathcal{R}_h^c = C \sum_{n \in \mathcal{I}_{\text{tr}}} \max_y (f(y, \mathbf{x}_n; p(\mathbf{Z}, \boldsymbol{\eta})) + \ell(y, y_n) - f(y_n, \mathbf{x}_n; p(\mathbf{Z}, \boldsymbol{\eta})))$; if $p$ is 2, the surrogate loss is the $\ell_2$-loss. For clarity, we consider the hinge loss. The non-negative cost function $\ell(y, y_n)$ (e.g., 0/1-cost) measures the cost of predicting $\mathbf{x}_n$ to be $y$ when its true label is $y_n$. $\mathcal{I}_{\text{tr}}$ is the index set of training data.

In order to robustly estimate the latent matrix $\mathbf{Z}$, we need a reasonable amount of data. Therefore, we also relate $\mathbf{Z}$ to the observed data $\mathbf{x}$ by defining a likelihood model to provide as much data as possible. Here, we define the linear-Gaussian likelihood model for real-valued data

$$p(\mathbf{x}_n | \mathbf{z}_n, \mathbf{W}, \sigma_{n0}^2) = \mathcal{N}(\mathbf{x}_n | \mathbf{W}\mathbf{z}_n^\top, \sigma_{n0}^2 I), \tag{8}$$

where $\mathbf{W}$ is a random loading matrix and $I$ is an identity matrix with appropriate dimensions. We assume $\mathbf{W}$ follows an independent Gaussian prior, i.e., $\pi(\mathbf{W}) = \prod_d \mathcal{N}(\mathbf{w}_d | 0, \sigma_0^2 I)$. Fig. 1 (a) shows the graphical structure of iLSVM. The hyperparameters $\sigma_0^2$ and $\sigma_{n0}^2$ can be set a priori or estimated from observed data (See Appendix A.2 for details).

**Testing**: to make prediction on test examples, we put both training and test data together to do the regularized Bayesian inference. For training data, we impose the above large-margin constraints because of the awareness of their true labels, while for test data, we do the inference without the large-margin constraints since we do not know their true labels. After inference, we make the prediction via the rule

$$y^* \stackrel{\text{def}}{=} \arg \max_y f(y, \mathbf{x}; p(\mathbf{Z}, \boldsymbol{\eta})). \tag{9}$$

The ability to generalize to test data relies on the fact that all the data examples share $\boldsymbol{\eta}$ and the IBP prior. We can also cast the problem as a transductive inference problem by imposing additional constraints on test data [17]. However, the resulting problem will be generally harder to solve.

### 3.3 Multi-Task Infinite Latent Support Vector Machines

Different from classification, which is typically formulated as a single learning task, multi-task learning aims to improve a set of related tasks through sharing statistical strength between these tasks, which are performed jointly. Many different approaches have been developed for multi-task learning (See [16] for a review). In particular, learning a common latent representation shared by all the related tasks has proven to be an effective way to capture task relationships [1, 3, 23]. Below, we present the multi-task infinite latent SVM (MT-iLSVM) for learning a common binary projection matrix $\mathbf{Z}$ to capture the relationships among multiple tasks. Similar as in iLSVM, we also put the IBP prior on $\mathbf{Z}$ to allow it to have an unbounded number of columns.

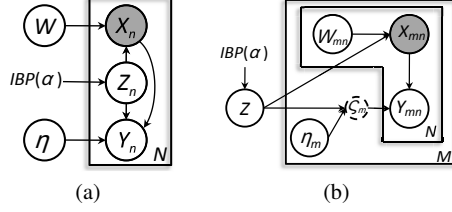

Figure 1: Graphical structures of (a) infinite latent SVM (iLSVM); and (b) multi-task infinite latent SVM (MT-iLSVM). For MT-iLSVM, the dashed nodes (i.e., $\varsigma_m$) are included to illustrate the task relatedness. We have omitted the priors on $\mathbf{W}$ and $\boldsymbol{\eta}$ for notation brevity.

(a)    (b)

Suppose we have $M$ related tasks. Let $\mathcal{D}_m = \{(\mathbf{x}_{mn}, y_{mn})\}_{n \in \mathcal{I}_{\text{tr}}^m}$ be the training data for task $m$. We consider binary classification tasks, where $\mathcal{Y}_m = \{+1, -1\}$. Extension to multi-way classification or regression tasks can be easily done. If the latent matrix $\mathbf{Z}$ is given, we define the latent discriminant function for task $m$ as

$$f_m(\mathbf{x}, \mathbf{Z}; \boldsymbol{\eta}_m) \stackrel{\text{def}}{=} (\mathbf{Z}\boldsymbol{\eta}_m)^\top \mathbf{x} = \boldsymbol{\eta}_m^\top (\mathbf{Z}^\top \mathbf{x}). \tag{10}$$

This definition provides two views of how the $M$ tasks get related. If we let $\varsigma_m = \mathbf{Z}\boldsymbol{\eta}_m$, then $\varsigma_m$ are the actual parameters of task $m$ and all $\varsigma_m$ in different tasks are coupled by sharing the same latent matrix $\mathbf{Z}$. Another view is that each task $m$ has its own parameters $\boldsymbol{\eta}_m$, but all the tasks share the same latent features $\mathbf{Z}^\top \mathbf{x}$, which is a projection of the input features $\mathbf{x}$ and $\mathbf{Z}$ is the latent projection matrix. As such, our method can be viewed as a nonparametric Bayesian treatment of alternating structure optimization (ASO) [1], which learns a single projection matrix with a pre-specified latent dimension. Moreover, different from [16], which learns a binary vector with known dimensionality to select features or kernels on $\mathbf{x}$, we learn an unbounded projection matrix $\mathbf{Z}$ using nonparametric Bayesian techniques.

As in iLSVM, we take the fully Bayeisan treatment (i.e., $\boldsymbol{\eta}_m$ are also random variables) and define the effective discriminant function for task $m$ as the expectation

$$f_m(\mathbf{x}; p(\mathbf{Z}, \boldsymbol{\eta})) \stackrel{\text{def}}{=} \mathbb{E}_{p(\mathbf{Z}, \boldsymbol{\eta})}[f_m(\mathbf{x}, \mathbf{Z}; \boldsymbol{\eta}_m)] = \mathbb{E}_{p(\mathbf{Z}, \boldsymbol{\eta})}[\mathbf{Z}\boldsymbol{\eta}_m]^\top \mathbf{x}. \tag{11}$$

Then, the prediction rule for task $m$ is naturally $y_m^* \stackrel{\text{def}}{=} \text{sign} f_m(\mathbf{x})$. Similarly, we do regularized Bayesian inference by imposing the following constraints and defining $U^{MT}(\boldsymbol{\xi}) \stackrel{\text{def}}{=} C \sum_{m, n \in \mathcal{I}_{\text{tr}}^m} \xi_{mn}$

$$\mathcal{P}_{\text{post}}^{MT}(\boldsymbol{\xi}) \stackrel{\text{def}}{=} \left\{ p(\mathbf{Z}, \boldsymbol{\eta}) \middle| \begin{array}{l} \forall m, \ \forall n \in \mathcal{I}_{\text{tr}}^m : \ y_{mn} \mathbb{E}_{p(\mathbf{Z}, \boldsymbol{\eta})}[\mathbf{Z}\boldsymbol{\eta}_m]^\top \mathbf{x}_{mn} \geq 1 - \xi_{mn} \\ \xi_{mn} \geq 0 \end{array} \right\}. \tag{12}$$

Similar as in iLSVM, minimizing $U^{MT}(\boldsymbol{\xi})$ is equivalent to minimizing the hinge-loss $\mathcal{R}_h^{MT}$ of the multiple binary prediction rules, where $\mathcal{R}_h^{MT} = C \sum_{m, n \in \mathcal{I}_{\text{tr}}^m} \max(0, 1 - y_{mn} \mathbb{E}_{p(\mathbf{Z}, \boldsymbol{\eta})}[\mathbf{Z}\boldsymbol{\eta}_m]^\top \mathbf{x}_{mn})$. Finally, to obtain more data to estimate the latent $\mathbf{Z}$, we also relate it to observed data by defining the likelihood model

$$p(\mathbf{x}_{mn}|\mathbf{w}_{mn}, \mathbf{Z}, \lambda_{mn}^2) = \mathcal{N}(\mathbf{x}_{mn}|\mathbf{Z}\mathbf{w}_{mn}, \lambda_{mn}^2 I), \tag{13}$$

where $\mathbf{w}_{mn}$ is a vector. We assume $\mathbf{W}$ has an independent prior $\pi(\mathbf{W}) = \prod_{mn} \mathcal{N}(\mathbf{w}_{mn}|0, \sigma_{m0}^2 I)$. Fig. 1 (b) illustrates the graphical structure of MT-iLSVM. For testing, we use the same strategy as in iLSVM to do Bayesian inference on both training and test data. The difference is that training data are subject to large-margin constraints, while test data are not. Similarly, the hyper-parameters $\sigma_{m0}^2$ and $\lambda_{mn}^2$ can be set a priori or estimated from data (See Appendix A.1 for details).

### 3.4 Inference with Truncated Mean-Field Constraints

We briefly discuss how to do regularized Bayesian inference (3) with the large-margin constraints for MT-iLSVM. For iLSVM, similar procedure applies. To make the problem easier to solve, we use the stick-breaking representation of IBP, which includes the auxiliary variables $\boldsymbol{\nu}$, and infer the posterior $p(\boldsymbol{\nu}, \mathbf{W}, \mathbf{Z}, \boldsymbol{\eta})$. Furthermore, we impose the truncated mean-field constraint that

$$p(\boldsymbol{\nu}, \mathbf{W}, \mathbf{Z}, \boldsymbol{\eta}) = p(\boldsymbol{\eta}) \prod_{k=1}^K \left( p(\nu_k|\boldsymbol{\gamma}_k) \prod_{d=1}^D p(z_{dk}|\psi_{dk}) \right) \prod_{mn} p(\mathbf{w}_{mn}|\Phi_{mn}, \sigma_{mn}^2 I), \tag{14}$$

where $K$ is the truncation level; $p(\mathbf{w}_{mn}|\Phi_{mn}, \sigma_{mn}^2 I) = \mathcal{N}(\mathbf{w}_{mn}|\Phi_{mn}, \sigma_{mn}^2 I)$; $p(z_{dk}|\psi_{dk}) =$ Bernoulli($\psi_{dk}$); and $p(\nu_k|\boldsymbol{\gamma}_k) = \text{Beta}(\gamma_{k1}, \gamma_{k2})$. We first turn the constrained problem to a problem of finding a stationary point using Lagrangian methods by introducing Lagrange multipliers $\boldsymbol{\omega}$, one for each large-margin constraint as defined in Eq. (12), and $\mathbf{u}$ for the nonnegativity constraints of $\boldsymbol{\xi}$. Let $L(p, \boldsymbol{\xi}, \boldsymbol{\omega}, \mathbf{u})$ be the Lagrangian functional. The inference procedure iteratively solves the following two steps (We defer the details to Appendix A.1):

**Infer $p(\boldsymbol{\nu})$, $p(\mathbf{W})$, and $p(\mathbf{Z})$:** for $p(\mathbf{W})$, since the prior is also normal, we can easily derive the update rules for $\Phi_{mn}$ and $\sigma^2_{mn}$. For $p(\boldsymbol{\nu})$, we have the same update rules as in [9]. We defer the details to Appendix A.1. Now, we focus on $p(\mathbf{Z})$ and provide insights on how the large-margin constraints regularize the procedure of inferring the latent matrix $\mathbf{Z}$. Since the large-margin constraints are linear of $p(\mathbf{Z})$, we can get the mean-field update equation as $\psi_{dk} = \frac{1}{1+e^{-\vartheta_{dk}}}$, where

$$\vartheta_{dk} = \sum_{j=1}^{k} \mathbb{E}_p[\log v_j] - \mathcal{L}_k^\nu - \sum_{mn} \frac{1}{2\lambda_{mn}^2}\Big( (K\sigma_{mn}^2 + (\phi_{mn}^k)^2) \tag{15}$$

$$-2x_{mn}^d \phi_{mn}^k + 2\sum_{j\neq k} \phi_{mn}^j \phi_{mn}^k \psi_{dj}\Big) + \sum_{m,n\in\mathcal{I}_{\text{tr}}^m} y_{mn}\mathbb{E}_p[\eta_{mk}]x_{mn}^d,$$

where $\mathcal{L}_k^\nu$ is an lower bound of $\mathbb{E}_p[\log(1 - \prod_{j=1}^{k} v_j)]$ (See Appendix A.1 for details). The last term of $\vartheta_{dk}$ is due to the large-margin posterior constraints as defined in Eq. (12).

**Infer $p(\boldsymbol{\eta})$ and solve for $\boldsymbol{\omega}$ and $\boldsymbol{\xi}$:** We optimize $L$ over $p(\boldsymbol{\eta})$ and can get $p(\boldsymbol{\eta}) = \prod_m p(\boldsymbol{\eta}_m)$, where

$$p(\boldsymbol{\eta}_m) \propto \pi(\boldsymbol{\eta}_m)\exp\{\boldsymbol{\eta}_m^\top \boldsymbol{\mu}_m\},$$

and $\boldsymbol{\mu}_m = \sum_{n\in\mathcal{I}_{\text{tr}}^m} y_{mn}\omega_{mn}(\boldsymbol{\psi}^\top \mathbf{x}_{mn})$. Here, we assume $\pi(\boldsymbol{\eta}_m)$ is standard normal. Then, we have $p(\boldsymbol{\eta}_m) = \mathcal{N}(\boldsymbol{\eta}_m|\boldsymbol{\mu}_m, I)$. Substituting the solution of $p(\boldsymbol{\eta})$ into $L$, we get $M$ independent dual problems

$$\max_{\boldsymbol{\omega}_m} \ -\frac{1}{2}\boldsymbol{\mu}_m^\top \boldsymbol{\mu}_m + \sum_{n\in\mathcal{I}_{\text{tr}}^m} \omega_{mn} \quad \text{s.t..}: \ 0 \leq \omega_{mn} \leq 1, \forall n \in \mathcal{I}_{\text{tr}}^m, \tag{16}$$

which (or its primal form) can be efficiently solved with a binary SVM solver, such as SVM-light.

# 4 Experiments

We present empirical results for both classification and multi-task learning. Our results demonstrate the merits inherited from both Bayesian nonparametrics and large-margin learning.

## 4.1 Multi-way Classification

We evaluate the infinite latent SVM (iLSVM) for classification on the real TRECVID2003 and Flickr image datasets, which have been extensively evaluated in the context of learning finite latent feature models [8]. TRECVID2003 consists of 1078 video key-frames, and each example has two types of features – 1894-dimension binary vector of text features and 165-dimension HSV color histogram. The Flickr image dataset consists of 3411 natural scene images about 13 types of animals (e.g., tiger, cat and etc.) downloaded from the Flickr website. Also, each example has two types of features, including 500-dimension SIFT bag-of-words and 634-dimension real-valued features (e.g., color histogram, edge direction histogram, and block-wise color moments). Here, we consider the real-valued features only by using normal distributions for $\mathbf{x}$.

We compare iLSVM with the large-margin Harmonium (MMH) [8], which was shown to outperform many other latent feature models [8], and two decoupled approaches – *EFH+SVM* and *IBP+SVM*. EFH+SVM uses the exponential family Harmonium (EFH) [27] to discover latent features and then learns a multi-way SVM classifier. IBP+SVM is similar, but uses an IBP factor analysis model [12] to discover latent features. As finite models, both MMH and EFH+SVM need to pre-specify the dimensionality of latent features. We report their results on classification accuracy and F1 score (i.e., the average F1 score over all possible classes) [32] achieved with the best dimensionality in Table 1. For iLSVM and IBP+SVM, we use the mean-field inference method and present the average performance with 5 randomly initialized runs (See Appendix A.2 for the algorithm and initialization details). We perform 5-fold cross-validation on training data to select hyperparameters, e.g., $\alpha$ and $C$ (we use the same procedure for MT-iLSVM). We can see that iLSVM can achieve comparable performance with the nearly optimal MMH, without needing to pre-specify the latent feature dimension[4], and is much better than the decoupled approaches (i.e., IBP+SVM and EFH+SVM).

## 4.2 Multi-task Learning
### 4.2.1 Description of the Data
**Scene and Yeast Data**: These datasets are from the UCI repository, and each data example has multiple labels. As in [23], we treat the multi-label classification as a multi-task learning problem,

Table 1: Classification accuracy and F1 scores on the TRECVID2003 and Flickr image datasets.

| Model | TRECVID2003 | | Flickr | |
|---|---|---|---|---|
| | Accuracy | F1 score | Accuracy | F1 score |
| EFH+SVM | $0.565 \pm 0.0$ | $0.427 \pm 0.0$ | $0.476 \pm 0.0$ | $0.461 \pm 0.0$ |
| MMH | $\mathbf{0.566} \pm 0.0$ | $0.430 \pm 0.0$ | $\mathbf{0.538} \pm 0.0$ | $\mathbf{0.512} \pm 0.0$ |
| IBP+SVM | $0.553 \pm 0.013$ | $0.397 \pm 0.030$ | $0.500 \pm 0.004$ | $0.477 \pm 0.009$ |
| iLSVM | $0.563 \pm 0.010$ | $\mathbf{0.448} \pm 0.011$ | $0.533 \pm 0.005$ | $0.510 \pm 0.010$ |

Table 2: Multi-label classification performance on Scene and Yeast datasets.

| Model | Yeast | | | Scene | | |
|---|---|---|---|---|---|---|
| | Acc | F1-Micro | F1-Macro | Acc | F1-Micro | F1-Macro |
| yaxue [23] | 0.5106 | 0.3897 | 0.4022 | 0.7765 | 0.2669 | 0.2816 |
| piyushrai-1 [23] | 0.5212 | 0.3631 | 0.3901 | 0.7756 | 0.3153 | 0.3242 |
| piyushrai-2 [23] | 0.5424 | 0.3946 | 0.4112 | 0.7911 | 0.3214 | 0.3226 |
| MT-IBP+SVM | $0.5475 \pm 0.005$ | $0.3910 \pm 0.006$ | $0.4345 \pm 0.007$ | $0.8590 \pm 0.002$ | $0.4880 \pm 0.012$ | $0.5147 \pm 0.018$ |
| MT-iLSVM | $\mathbf{0.5792} \pm 0.003$ | $\mathbf{0.4258} \pm 0.005$ | $\mathbf{0.4742} \pm 0.008$ | $\mathbf{0.8752} \pm 0.004$ | $\mathbf{0.5834} \pm 0.026$ | $\mathbf{0.6148} \pm 0.020$ |

where each label assignment is treated as a binary classification task. The Yeast dataset consists of 1500 training and 917 test examples, each having 103 features, and the number of labels (or tasks) per example is 14. The Scene dataset consists 1211 training and 1196 test examples, each having 294 features, and the number of labels (or tasks) per example for this dataset is 6.

**School Data**: This dataset comes from the Inner London Education Authority and has been used to study the effectiveness of schools. It consists of examination records from 139 secondary schools in years 1985, 1986 and 1987. It is a random $50\%$ sample with 15362 students. The dataset is publicly available and has been extensively evaluated in various multi-task learning methods [4, 7, 30], where each task is defined as predicting the exam scores of students belonging to a specific school based on four student-dependent features (year of the exam, gender, VR band and ethnic group) and four school-dependent features (percentage of students eligible for free school meals, percentage of students in VR band 1, school gender and school denomination). In order to compare with the above methods, we follow the same setup described in [3, 4] and similarly we create dummy variables for those features that are categorical forming a total of 19 student-dependent features and 8 school-dependent features. We use the same 10 random splits[5] of the data, so that $75\%$ of the examples from each school (task) belong to the training set and $25\%$ to the test set. On average, the training set includes about 80 students per school and the test set about 30 students per school.

### 4.2.2 Results

**Scene and Yeast Data**: We compare with the closely related nonparametric Bayesian methods [23, 28], which were shown to outperform the independent Bayesian logistic regression and a single-task pooling approach [23], and a decoupled method *MT-IBP+SVM*[6] that uses IBP factor analysis model to find shared latent features among multiple tasks and then builds separate SVM classifiers for different tasks. For MT-iLSVM and MT-IBP+SVM, we use the mean-field inference method in Sec 3.4 and report the average performance with 5 randomly initialized runs (See Appendix A.1 for initialization details). For comparison with [23, 28], we use the overall classification accuracy, F1-Macro and F1-Micro as performance measures. Table 2 shows the results. We can see that the large-margin MT-iLSVM performs much better than other nonparametric Bayesian methods and MT-IBP+SVM, which separates the inference of latent features from learning the classifiers.

**School Data**: We use the percentage of explained variance [4] as the measure of the regression performance, which is defined as the total variance of the data minus the sum-squared error on the test set as a percentage of the total variance. Since we use the same settings, we can compare with the state-of-the-art results of Bayesian multi-task learning (BMTL) [4], multi-task Gaussian processes (MTGP) [7], convex multi-task relationship learning (MTRL) [30], and single-task learning (STL) as reported in [7, 30]. For MT-iLSVM and MT-IBP+SVM, we also report the results achieved by using both the latent features (i.e., $\mathbf{Z}^\top \mathbf{x}$) and the original input features $\mathbf{x}$ through vector concatenation, and we denote the corresponding methods by *MT-iLSVM$^f$* and *MT-IBP+SVM$^f$*, respectively. From

Table 3: Percentage of explained variance by various models on the School dataset.

| STL | BMTL | MTGP | MTRL | MT-IBP+SVM | MT-iLSVM | MT-IBP+SVM$^f$ | MT-iLSVM$^f$ |
|---|---|---|---|---|---|---|---|
| $23.5 \pm 1.9$ | $29.5 \pm 0.4$ | $29.2 \pm 1.6$ | $29.9 \pm 1.8$ | $20.0 \pm 2.9$ | $30.9 \pm 1.2$ | $28.5 \pm 1.6$ | $\mathbf{31.7 \pm 1.1}$ |

Table 4: Percentage of explained variance and running time by MT-iLSVM with various training sizes.

| | 50% | 60% | 70% | 80% | 90% | 100% |
|---|---|---|---|---|---|---|
| explained variance (%) | $25.8 \pm 0.4$ | $27.3 \pm 0.7$ | $29.6 \pm 0.4$ | $30.0 \pm 0.5$ | $30.8 \pm 0.4$ | $30.9 \pm 1.2$ |
| running time (s) | $370.3 \pm 32.5$ | $455.9 \pm 18.6$ | $492.6 \pm 33.2$ | $600.1 \pm 50.2$ | $777.6 \pm 73.4$ | $918.9 \pm 96.5$ |

the results in Table 3, we can see that the multi-task latent SVM (i.e., MT-iLSVM) achieves better results than the existing methods that have been tested in previous studies. Again, the joint MT-iLSVM performs much better than the decoupled method MT-IBP+SVM, which separates the latent feature inference from the training of large-margin classifiers. Finally, using both latent features and the original input features can boost the performance slightly for MT-iLSVM, while much more significantly for the decoupled MT-IBP+SVM.

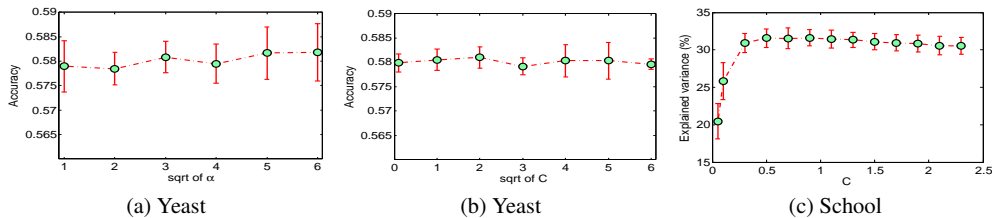

(a) Yeast  (b) Yeast  (c) School

Figure 2: Sensitivity study of MT-iLSVM: (a) classification accuracy with different $\alpha$; (b) classification accuracy with different $C$; and (c) percentage of explained variance with different $C$.

## 4.3 Sensitivity Analysis

Figure 2 shows how the performance of MT-iLSVM changes against the hyper-parameter $\alpha$ and regularization constant $C$ on Yeast and School datasets. We can see that on the Yeast dataset, MT-iLSVM is insensitive to $\alpha$ and $C$. For the School dataset, MT-iLSVM is stable when $C$ is set between 0.3 and 1. MT-iLSVM is insensitive to $\alpha$ on the School data too, which is omitted to save space.

Table 4 shows how the training size affects the performance and running time of MT-iLSVM on the School dataset. We use the first $b\%$ ($b = 50, 60, 70, 80, 90, 100$) of the training data in each of the 10 random splits as training set and use the corresponding test data as test set. We can see that as training size increases, the performance and running time generally increase; and MT-iLSVM achieves the state-of-art performance when using about 70% training data. From the running time, we can also see that MT-iLSVM is generally quite efficient by using mean-field inference.

Finally, we investigate how the performance of MT-iLSVM changes against the hyperparameters $\sigma_{m0}^2$ and $\lambda_{mn}^2$. We initially set $\sigma_{m0}^2 = 1$ and compute $\lambda_{mn}^2$ from observed data. If we further estimate them by maximizing the objective function, the performance does not change much ($\pm 0.3\%$ for average explained variance on the School dataset). We have similar observations for iLSVM.

## 5 Conclusions and Future Work

We first present a general framework for doing regularized Bayesian inference subject to appropriate constraints, which are imposed directly on the posterior distributions. Then, we particularly concentrate on developing two nonparametric Bayesian models to learn predictive latent features for classification and multi-task learning, respectively, by exploring the large-margin principle to define posterior constraints. Both models allow the latent dimension to be automatically resolved from the data. The empirical results on several real datasets appear to demonstrate that our methods inherit the merits from both Bayesian nonparametrics and large-margin learning.

Regularized Bayesian inference offers a general framework for considering posterior regularization in performing nonparametric Bayesian inference. For future work, we plan to study other posterior regularization beyond the large-margin constraints, such as posterior constraints defined on manifold structures [14], and investigate how posterior regularization can be used in other interesting nonparametric Bayesian models [5, 26].

**Acknowledgments**

This work was done when JZ was a post-doc fellow in CMU. JZ is supported by National Key Project for Basic Research of China (No. 2012CB316300) and the National Natural Science Foundation of China (No. 60805023). EX is supported by AFOSR FA95501010247, ONR N000140910758, NSF Career DBI-0546594 and Alfred P. Sloan Research Fellowship.

## Footnotes

[1]Real-valued features can be easily considered as in [12].

[2]We can consider the input features $\mathbf{x}$ or its certain statistics in combination with the latent features $\mathbf{z}$ to define a classifier boundary, by simply concatenating them in the subvectors.

[3]Although other choices such as taking the mode are possible, our choice could lead to a computationally easy problem because expectation is a linear functional of the distribution under which the expectation is taken. Moreover, expectation can be more robust than taking the mode [18], and it has been used in [31, 32].

[4]We set the truncation level to 300, which is large enough.

[5]Available at: http://ttic.uchicago.edu/~argyriou/code/index.html

[6]This decoupled approach is in fact an one-iteration MT-iLSVM, where we first infer the shared latent matrix $\mathbf{Z}$ and then learn an SVM classifier for each task.

# References

[1] R. Ando and T. Zhang. A framework for learning predictive structures from multiple tasks and unlabeled data. *JMLR*, (6):1817–1853, 2005.

[2] C.E. Antoniak. Mixture of Dirichlet process with applications to Bayesian nonparametric problems. *Annals of Stats*, (273):1152–1174, 1974.

[3] A. Argyriou, T. Evgeniou, and M. Pontil. Convex multi-task feature learning. In *NIPS*, 2007.

[4] B. Bakker and T. Heskes. Task clustering and gating for Bayesian multitask learning. *JMLR*, (4):83–99, 2003.

[5] M.J. Beal, Z. Ghahramani, and C.E. Rasmussen. The infinite hidden Markov model. In *NIPS*, 2002.

[6] K. Bellare, G. Druck, and A. McCallum. Alternating projections for learning with expectation constraints. In *UAI*, 2009.

[7] E. Bonilla, K.M.A. Chai, and C. Williams. Multi-task Gaussian process prediction. In *NIPS*, 2008.

[8] N. Chen, J. Zhu, and E.P. Xing. Predictive subspace learning for multiview data: a large margin approach. In *NIPS*, 2010.

[9] F. Doshi-Velez, K. Miller, J. Van Gael, and Y.W. Teh. Variational inference for the Indian buffet process. In *AISTATS*, 2009.

[10] D. Dunson and S. Peddada. Bayesian nonparametric inferences on stochastic ordering. *ISDS Discussion Paper*, 2, 2007.

[11] K. Ganchev, J. Graca, J. Gillenwater, and B. Taskar. Posterior regularization for structured latent variable models. *JMLR*, (11):2001–2094, 2010.

[12] T.L. Griffiths and Z. Ghahramani. Infinite latent feature models and the Indian buffet process. In *NIPS*, 2006.

[13] D. Hoff. Bayesian methods for partial stochastic orderings. *Biometrika*, 90:303–317, 2003.

[14] S. Huh and S. Fienberg. Discriminative topic modeling based on manifold learning. In *KDD*, 2010.

[15] T. Jaakkola, M. Meila, and T. Jebara. Maximum entropy discrimination. In *NIPS*, 1999.

[16] T. Jebara. Multitask sparsity via maximum entropy discrimination. *JMLR*, (12):75–110, 2011.

[17] T. Joachims. Transductive inference for text classification using support vector machines. In *ICML*, 1999.

[18] M. E. Khan, B. Marlin, G. Bouchard, and K. Murphy. Variational bounds for mixed-data factor analysis. In *NIPS*, 2010.

[19] P. Liang, M. Jordan, and D. Klein. Learning from measurements in exponential families. In *ICML*, 2009.

[20] S.N. MacEachern. Dependent nonparametric process. In *the Section on Bayesian Statistical Science of ASA*, 1999.

[21] G. Mann and A. McCallum. Generalized expectation criteria for semi-supervised learning with weakly labeled data. *JMLR*, (11):955–984, 2010.

[22] K. Miller, T. Griffiths, and M. Jordan. Nonparametric latent feature models for link prediction. In *NIPS*, 2009.

[23] P. Rai and H. Daume III. Infinite predictor subspace models for multitask learning. In *AISTATS*, 2010.

[24] C.E. Rasmussen and Z. Ghahramani. Infinite mixtures of Gaussian process experts. In *NIPS*, 2002.

[25] Y.W. Teh, D. Gorur, and Z. Ghahramani. Stick-breaking construction of the Indian buffet process. In *AISTATS*, 2007.

[26] Y.W. Teh, M. Jordan, M. Beal, and D. Blei. Hierarchical Dirichlet process. *JASA*, 101(476):1566–1581, 2006.

[27] M. Welling, M. Rosen-Zvi, and G. Hinton. Exponential family harmoniums with an application to information retrieval. In *NIPS*, 2004.

[28] Y. Xue, D. Dunson, and L. Carin. The matrix stick-breaking process for flexible multi-task learning. In *ICML*, 2007.

[29] A. Zellner. Optimal information processing and Bayes' theorem. *American Statistician*, 42:278–280, 1988.

[30] Y. Zhang and D.Y. Yeung. A convex formulation for learning task relationships in multi-task learning. In *UAI*, 2010.

[31] J. Zhu, A. Ahmed, and E.P. Xing. MedLDA: Maximum margin supervised topic models for regression and classification. In *ICML*, 2009.

[32] J. Zhu, N. Chen, and E.P. Xing. Infinite SVM: a Dirichlet process mixture of large-margin kernel machines. In *ICML*, 2011.

